# Asymptotic Convergence of Backpropagation: Numerical Experiments

**Subutai Ahmad**
ICSI
1947 Center St.
Berkeley, CA 94704

**Gerald Tesauro**
IBM Watson Labs.
P. O. Box 704
Yorktown Heights, NY
10598

**Yu He**
Dept. of Physics
Ohio State Univ.
Columbus, OH 43212

## ABSTRACT

We have calculated, both analytically and in simulations, the rate of convergence at long times in the backpropagation learning algorithm for networks with and without hidden units. Our basic finding for units using the standard sigmoid transfer function is $1/t$ convergence of the error for large $t$, with at most logarithmic corrections for networks with hidden units. Other transfer functions may lead to a slower polynomial rate of convergence. Our analytic calculations were presented in (Tesauro, He & Ahamd, 1989). Here we focus in more detail on our empirical measurements of the convergence rate in numerical simulations, which confirm our analytic results.

## 1   INTRODUCTION

Backpropagation is a popular learning algorithm for multilayer neural networks which minimizes a global error function by gradient descent (Werbos, 1974: Parker, 1985; LeCun, 1985; Rumelhart, Hinton & Williams, 1986). In this paper, we examine the rate of convergence of backpropagation late in learning when all of the errors are small. In this limit, the learning equations become more amenable to analytic study. By expanding in the small differences between the desired and actual output states, and retaining only the dominant terms, one can explicitly solve for the leading-order behavior of the weights as a function of time. This is true both for

single-layer networks, and for multilayer networks containing hidden units. We confirm our analysis by empirical measurements of the convergence rate in numerical simulations.

In gradient-descent learning, one minimizes an error function $E$ according to:

$$\Delta \vec{w} = -\epsilon \frac{\partial E}{\partial \vec{w}} \qquad (1)$$

where $\Delta \vec{w}$ is the change in the weight vector at each time step, and the learning rate $\epsilon$ is a small numerical constant. The convergence of equation 1 for single-layer networks with general error functions and transfer functions is studied in section 2. In section 3, we examine two standard modifications of gradient-descent: the use of a "margin" variable for turning off the error backpropagation, and the inclusion of a "momentum" term in the learning equation. In section 4 we consider networks with hidden units, and in the final section we summarize our results and discuss possible extensions in future work.

## 2   CONVERGENCE IN SINGLE-LAYER NETWORKS

The input-output relationship for single-layer networks takes the form:

$$y_p = g(\vec{w} \cdot \vec{x}_p) \qquad (2)$$

where $\vec{x}_p$ represents the state of the input units for pattern $p$, $\vec{w}$ is the real-valued weight vector of the network, $g$ is the input-output transfer function (for the moment unspecified), and $y_p$ is the output state for pattern $p$. We assume that the transfer function approaches 0 for large negative inputs and 1 for large positive inputs.

For convenience of analysis, we rewrite equation 1 for continuous time as:

$$\dot{\vec{w}} = -\epsilon \sum_p \frac{\partial E_p}{\partial \vec{w}} = -\epsilon \sum_p \frac{\partial E_p}{\partial y_p} \frac{\partial y_p}{\partial \vec{w}} = -\epsilon \sum_p \frac{\partial E_p}{\partial y_p} g'(h_p) \vec{x}_p \qquad (3)$$

where $E_p$ is the individual error for pattern $p$, $h_p = \vec{w} \cdot \vec{x}_p$ is the total input activation of the output unit for pattern $p$, and the summation over $p$ is for an arbitrary subset of the possible training patterns. $E_p$ is a function of the difference between the actual output $y_p$ and the desired output $d_p$ for pattern $p$. Examples of common error functions are the quadratic error $E_p = (y_p - d_p)^2$ and the "cross-entropy" error (Hinton, 1987) $E_p = d_p \log y_p + (1 - d_p) \log(1 - y_p)$.

Instead of solving equation 3 for the weights directly, it is more convenient to work with the outputs $y_p$. The outputs evolve according to:

$$\dot{y}_p = -\epsilon g'(h_p) \sum_q \frac{\partial E_q}{\partial y_q} g'(h_q) \vec{x}_q \cdot \vec{x}_p \qquad (4)$$

Let us now consider the situation late in learning when the output states are approaching the desired values. We define new variables $\eta_p = y_p - d_p$, and assume

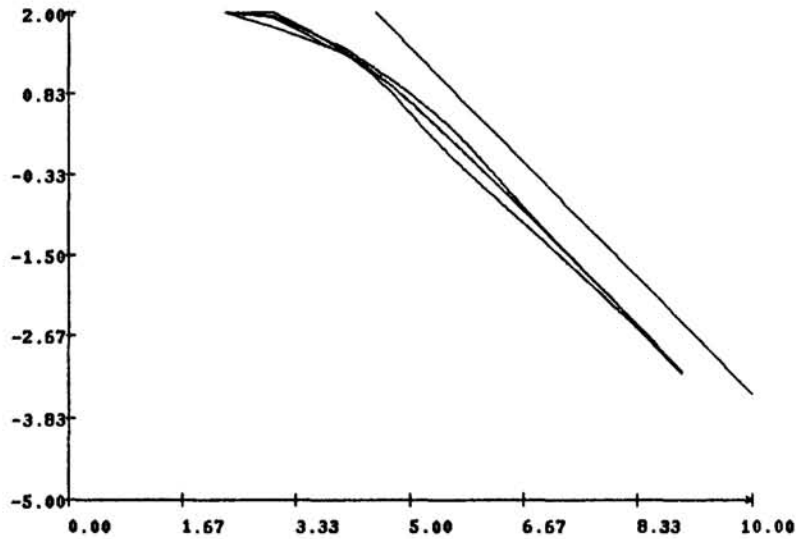

**Figure 1:** Plots of ln(error) vs. ln(epochs) for single-layer networks learning the majority function using standard backpropagation without momentum. Four different learning runs starting from different random initial weights are shown. In each case, the asymptotic behavior is approximately $E \sim 1/t$, as seen by comparison with a reference line of slope -1.

that $\eta_p$ is small for all $p$. For reasonable error functions, the individual errors $E_p$ will go to zero as some power of $\eta_p$, i.e., $E_p \sim \eta_p^\gamma$. (For the quadratic error, $\gamma = 2$, and for the cross-entropy error, $\gamma = 1$.) Similarly, the slope of the transfer function should approach zero as the output state approaches 1 or 0, and for reasonable transfer functions, this will again follow a power law, i.e., $g'(h_p) \sim \eta_p^\beta$. Using the definitions of $\eta$, $\gamma$ and $\beta$, equation 4 becomes:

$$\dot{\eta}_p \sim |\eta_p|^\beta \sum_q \eta_q^{\gamma-1} |\eta_q|^\beta \, \vec{x}_q \cdot \vec{x}_p + \text{higher order} \tag{5}$$

The absolute value appears because $g$ is a non-decreasing function. Let $\eta_r$ be the slowest to approach zero among all the $\eta_p$'s. We then have for $\eta_r$:

$$\dot{\eta}_r \sim \eta_r^{2\beta+\gamma-1} \tag{6}$$

Upon integrating we obtain

$$\eta_r \sim t^{-1/(2\beta+\gamma-2)} \; ; \; E \sim \eta_r^\gamma \sim t^{-\gamma/(2\beta+\gamma-2)} \tag{7}$$

When $\beta = 1$, i.e., $g' \sim \eta$, the error function approaches zero like $1/t$, independent of $\gamma$. Since $\beta = 1$ for the standard sigmoid function $g(x) = (1 + e^{-x})^{-1}$, one expects to see $1/t$ behavior in the error function in this case. This behavior was in fact first

seen in the numerical experiments of (Ahmad, 1988; Ahmad & Tesauro, 1988). The behavior was obtained at relatively small $t$, about 20 cycles through the training set. Figure 1 illustrates this behavior for single-layer networks learning a data set containing 200 randomly chosen instances of the majority function. In each case, the behavior at long times in this plot is approximately a straight line, indicating power-law decrease of the error. The slopes are in each case within a few percent of the theoretically predicted value of -1.

It turns out that $\beta = 1$ gives the fastest possible convergence of the error function. This is because $\beta < 1$ yields transfer functions which do not saturate at finite values, and thus are not allowed, while $\beta > 1$ yields slower convergence. For example, if we take the transfer function to be $g(x) = 0.5[1 + (2/\pi)\tan^{-1} x]$, then $\beta = 2$. In this case, the error function will go to zero as $E \sim t^{-\gamma/(\gamma+2)}$. In particular, when $\gamma = 2$, $E \sim 1/\sqrt{t}$.

## 3   MODIFICATIONS OF GRADIENT DESCENT

One common modification to strict gradient-descent is the use of a "margin" variable $\mu$ such that, if the difference between network output and teacher signal is smaller than $\mu$, no error is backpropagated. This is meant to prevent the network from devoting resources to making its output arbitrarily close to the teacher signal, which is usually unnecessary. It is clear from the structure of equations 5, 6 that the margin will not affect the basic $1/t$ error convergence, except in a rather trivial way. When a margin is employed, certain driving terms on the right-hand side of equation 5 will be set to zero as soon as they become small enough. However, as long as *some* non-zero driving terms are present, the basic polynomial solution of equation 7 will be unaltered. Of course, when *all* the driving terms disappear because they are all smaller than the margin, the network will stop learning, and the error will remain constant at some positive value. Thus the prediced behavior is $1/t$ decrease in the error followed eventually by a rapid transition to constant non-zero error. This agrees with what is seen numerically in Figure 2.

Another popular generalization of equation 1 includes a "momentum" term:

$$\Delta\vec{w}(t) = -\epsilon\,\frac{\partial E}{\partial\vec{w}}(t) + \alpha\Delta\vec{w}(t-1) \tag{8}$$

In continuous time, this takes the form:

$$\alpha\ddot{\vec{w}} + (1-\alpha)\dot{\vec{w}} = -\epsilon\,\frac{\partial E}{\partial\vec{w}} \tag{9}$$

Turning this into an equation for the evolution of outputs gives:

$$\alpha\ddot{y}_p - \alpha g''(h_p)[\frac{\dot{y}_p}{g'(h_p)}]^2 + (1-\alpha)\dot{y}_p = -\epsilon g'(h_p)\sum_q \frac{\partial E_q}{\partial y_q}g'(h_q)\vec{x}_q \cdot \vec{x}_p \tag{10}$$

Once again, exapanding $y_p$, $E_p$ and $g'$ in small $\eta_p$ yields a second-order differential equation for $\eta_p$ in terms of a sum over other $\eta_q$. As in equation 6, the sum will be

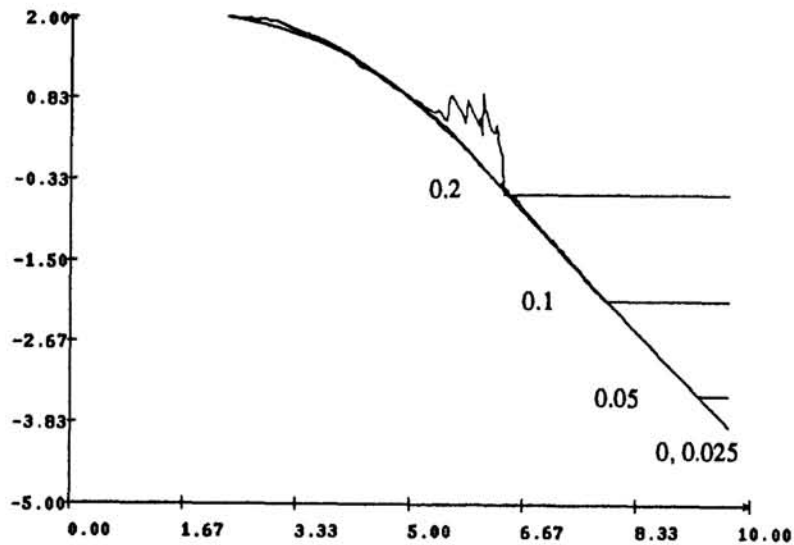

**Figure 2:** Plot of ln(error) vs. ln(epochs) for various values of margin variable $\mu$ as indicated. In each case there is a $1/t$ decrease in the error followed by a sudden transition to constant error. This transition occurs earlier for larger values of $\mu$.

controlled by some dominant term $r$, and the equation for this term is:

$$C_1\ddot{\eta}_r + C_2\eta_r^{-1}\dot{\eta}_r^2 + C_3\dot{\eta}_r \sim \eta_r^{2\beta+\gamma-1} \tag{11}$$

where $C_1, C_2$ and $C_3$ are numerical constants. For polynomial solutions $\eta_r \sim t^z$, the first two terms are of order $t^{z-2}$, and can be neglected relative to the third term which is of order $t^{z-1}$. The resulting equation thus has exactly the same form as in the zero momentum case of section 2, and therefore the rate of convergence is the same as in equation 7. This is demonstrated numerically in Figure 3. We can see that the error behaves as $1/t$ for large $t$ regardless of the value of momentum constant $\alpha$. Furthermore, although it is not required by the analytic theory, the numerical prefactor appears to be the same in each case.

Finally, we have also considered the effect on convergence of schemes for adaptively altering the learning rate constant $\epsilon$. It was shown analytically in (Tesauro, He & Ahmad, 1989) that for the scheme proposed by Jacobs (1988), in which the learning rate could in principle increase linearly with time, the error would decrease as $1/t^2$ for sigmoid units, instead of the $1/t$ result for fixed $\epsilon$.

## 4  CONVERGENCE IN NETWORKS WITH HIDDEN UNITS

We now consider networks with a single hidden layer. In (Tesauro, He & Ahmad, 1989), it was shown that if the hidden units saturate late in learning, then the convergence rate is no different from the single-layer rate. This should be typical

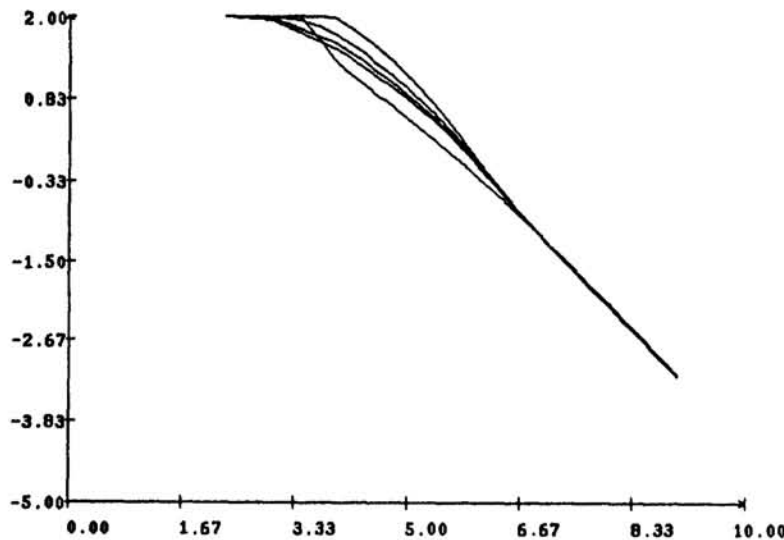

**Figure 3:** Plot of ln(error) vs. ln(epochs) for single-layer networks learning the majority function, with momentum constant $\alpha = 0, 0.25, 0.5, 0.75, 0.99$. Each run starts from the same random initial weights. Asymptotic $1/t$ behavior is obtained in each case, with the same numerical prefactor.

of what usually happens. However, assuming for purposes of argument that the hidden units do not saturate, when one goes through a small $\eta$ expansion of the learning equation, one obtains a coupled system of equations of the following form:

$$\dot{\eta} \sim \eta^{2\beta+\gamma-1}[1 + \Omega^2] \tag{12}$$

$$\dot{\Omega} \sim \eta^{\gamma+\beta-1} \tag{13}$$

where $\Omega$ represents the magnitude of the second layer weights, and for convenience all indices have been suppressed and all terms of order 1 have been written simply as 1.

For $\beta > 1$, this system has polynomial solutions of the form $\eta \sim t^z$, $\Omega \sim t^\lambda$, with $z = -3/(3\gamma + 4\beta - 4)$ and $\lambda = z(\gamma + \beta - 1) - 1$. It is interesting to note that these solutions converge slightly faster than in the single-layer case. For example, with $\gamma = 2$ and $\beta = 2$, $\eta \sim t^{-3/10}$ in the multilayer case, but as shown previously, $\eta$ goes to zero only as $t^{-1/4}$ in the single-layer case. We emphasize that this slight speed-up will only be obtained when the hidden unit states do not saturate. To the extent that the hidden units saturate and their slopes become small, the convergence rate will return to the single-layer rate.

When $\beta = 1$ the above polynomial solution is not possible. Instead, one can verify that the following is a self-consistent leading order solution to equations 12, 13:

$$\eta \sim t^{-1/\gamma} ln^{-2/3\gamma} t \tag{14}$$

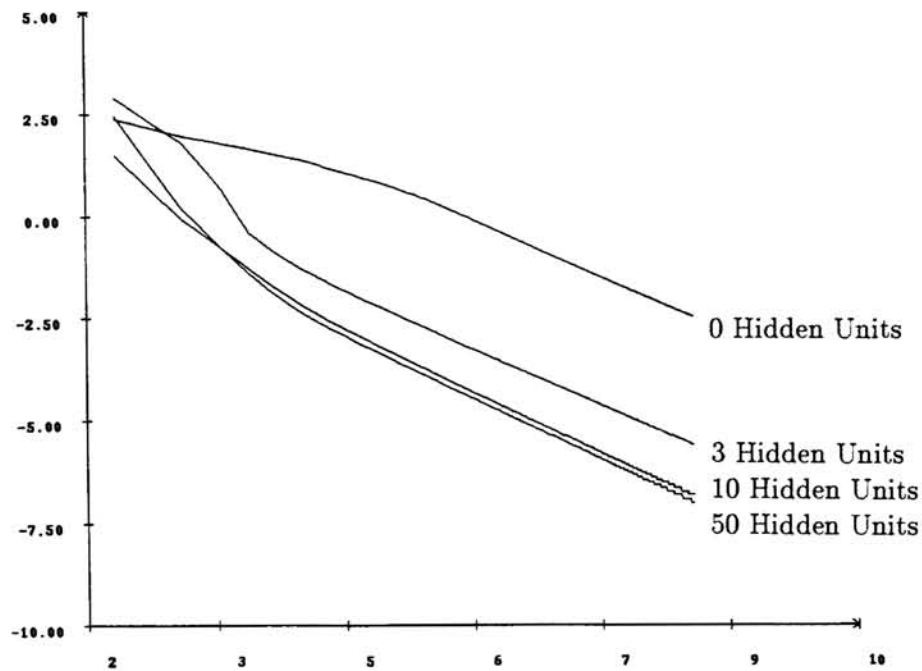

**Figure 4:** Plot of ln(error) vs. ln(epochs) for networks with varying numbers of hidden units (as indicated) learning majority function data set. Approximate $1/t$ behavior is obtained in each case.

$$\Omega \sim ln^{1/3}t \qquad (15)$$

Recall that in the single-layer case, $\eta \sim t^{-1/\gamma}$. Therefore, the effect of multiple layers could provide at most only a logarithmic speed-up of convergence when the hidden units do not saturate. For practical purposes, then, we expect the convergence of networks with hidden units to be no different empirically from networks without hidden units. This is in fact what our simulations find, as illustrated in Figure 4.

## 5   DISCUSSION

We have obtained results for the asymptotic convergence of gradient-descent learning which are valid for a wide variety of error functions and transfer functions. We typically expect the same rate of convergence to be obtained regardless of whether or not the network has hidden units. However, it may be possible to obtain a slight polynomial speed-up when $\beta > 1$ or a logarithmic speed-up when $\beta = 1$. We point out that in all cases, the sigmoid provides the maximum possible convergence rate, and is therefore a "good" transfer function to use in that sense.

We have not attempted analysis of networks with multiple layers of hidden units; however, the analysis of (Tesauro, He & Ahmad, 1989) suggests that, to the extent that the hidden unit states saturate and the $g'$ factors vanish, the rate of convergence would be no different even in networks with arbitrary numbers of hidden layers.

Another important finding is that the expected rate of convergence does not depend on the use of all $2^n$ input patterns in the training set. The same behavior should

be seen for general subsets of training data. This is also in agreement with our numerical results, and with the results of (Ahamd, 1988; Ahmand & Tesauro, 1988).

In conclusion, a combination of analysis and numerical simulations has led to insight into the late stages of gradient-descent learning. It might also be possible to extend our approach to times earlier in the learning process, when not all of the errors are small. One might also be able to analyze the numbers, sizes and shapes of the basins of attraction for gradient-descent learning in feed-forward networks. Another important issue is the behavior of the generalization performance, i.e., the error on a set of test patterns not used in training, which was not addressed in this paper. Finally, our analysis might provide insight into the development of new algorithms which might scale more favorably than backpropagation.

## References

S. Ahmad. (1988) A study of scaling and generalization in neural networks. Master's Thesis, Univ. of Illinois at Urbana-Champaign, Dept. of Computer Science.

S. Ahmad & G. Tesauro. (1988) Scaling and generalization in neural networks: a case study. In D. S. Touretzky et al. (eds.), *Proceedings of the 1988 Connectionist Models Summer School*, 3-10. San Mateo, CA: Morgan Kaufmann.

G. E. Hinton. (1987) Connectionist learning procedures. Technical Report No. CMU-CS-87-115, Dept. of Computer Science, Carnegie-Mellon University.

R. A. Jacobs. (1988) Increased rates of convergence through learning rate adaptation. *Neural Networks* 1:295-307.

Y. Le Cun. (1985) A learning procedure for asymmetric network. *Proceedings of Cognitiva (Paris)* 85:599-604.

D. B. Parker. (1985) Learning-logic. Technical Report No. TR-47, MIT Center for Computational Research in Economics and Management Science.

D. E. Rumelhart, G. E. Hinton, & R. J. Williams. (1986) Learning representations by back-propagating errors. *Nature* 323:533-536.

G. Tesauro, Y. He & S. Ahmad. (1989) Asymptotic convergence of backpropagation. *Neural Computation* 1:382-391.

P. Werbos. (1974) Ph. D. Thesis, Harvard University.
